# Learning Spectral Clustering

**Francis R. Bach**
Computer Science
University of California
Berkeley, CA 94720
*fbach@cs.berkeley.edu*

**Michael I. Jordan**
Computer Science and Statistics
University of California
Berkeley, CA 94720
*jordan@cs.berkeley.edu*

## Abstract

Spectral clustering refers to a class of techniques which rely on the eigen-structure of a similarity matrix to partition points into disjoint clusters with points in the same cluster having high similarity and points in different clusters having low similarity. In this paper, we derive a new cost function for spectral clustering based on a measure of error between a given partition and a solution of the spectral relaxation of a minimum normalized cut problem. Minimizing this cost function with respect to the partition leads to a new spectral clustering algorithm. Minimizing with respect to the similarity matrix leads to an algorithm for learning the similarity matrix. We develop a tractable approximation of our cost function that is based on the power method of computing eigenvectors.

## 1 Introduction

Spectral clustering has many applications in machine learning, exploratory data analysis, computer vision and speech processing. Most techniques explicitly or implicitly assume a metric or a similarity structure over the space of configurations, which is then used by clustering algorithms. The success of such algorithms depends heavily on the choice of the metric, but this choice is generally not treated as part of the learning problem. Thus, time-consuming manual feature selection and weighting is often a necessary precursor to the use of spectral methods.

Several recent papers have considered ways to alleviate this burden by incorporating prior knowledge into the metric, either in the setting of $K$-means clustering [1, 2] or spectral clustering [3, 4]. In this paper, we consider a complementary approach, providing a general framework for learning the similarity matrix for spectral clustering from examples. We assume that we are given sample data with known partitions and are asked to build similarity matrices that will lead to these partitions when spectral clustering is performed. This problem is motivated by the availability of such datasets for at least two domains of application: in vision and image segmentation, a hand-segmented dataset is now available [5], while for the blind separation of speech signals via partitioning of the time-frequency plane [6], training examples can be created by mixing previously captured signals.

Another important motivation for our work is the need to develop spectral clustering methods that are robust to irrelevant features. Indeed, as we show in Section 4.2, the performance of current spectral methods can degrade dramatically in the presence of such irrelevant features. By using our learning algorithm to learn a diagonally-scaled Gaussian kernel

for generating the affinity matrix, we obtain an algorithm that is significantly more robust.

Our work is based on a new cost function $J(W, e)$ that characterizes how close the eigenstructure of a similarity matrix $W$ is to a partition $e$. We derive this cost function in Section 2. As we show in Section 2.3, minimizing $J$ with respect to $e$ leads to a new clustering algorithm that takes the form of a weighted $K$-means algorithm. Minimizing $J$ with respect to $W$ yields an algorithm for learning the similarity matrix, as we show in Section 4. Section 3 provides foundational material on the approximation of the eigensubspace of a symmetric matrix that is needed for Section 4.

## 2 Spectral clustering and normalized cuts

Given a dataset $\mathcal{I}$ of $P$ points in a space $\mathcal{X}$ and a $P \times P$ "similarity matrix" (or "affinity matrix") $W$ that measures the similarity between the $P$ points ($W_{pp'}$ is large when points indexed by $p$ and $p'$ are likely to be in the same cluster), the goal of clustering is to organize the dataset into disjoint subsets with high intra-cluster similarity and low inter-cluster similarity. Throughout this paper we always assume that the elements of $W$ are non-negative ($W \geqslant 0$) and that $W$ is symmetric ($W = W^\top$).

Let $D$ denote the diagonal matrix whose $i$-th diagonal element is the sum of the elements in the $i$-th row of $W$, i.e., $D = \operatorname{diag}(W1)$, where $1$ is defined as the vector in $\mathbb{R}^P$ composed of ones. There are different variants of spectral clustering. In this paper we focus on the task of minimizing "normalized cuts." The classical relaxation of this NP-hard problem [7, 8, 9] leads to an eigenvalue problem. In this section we show that the problem of finding a solution to the original problem that is closest to the relaxed solution can be solved by a weighted $K$-means algorithm.

### 2.1 Normalized cut and graph partitioning

The clustering problem is usually defined in terms of a complete graph with vertices $V = \{1, ..., P\}$ and an affinity matrix with weights $W_{pp'}$, for $p, p' \in V$. We wish to find $R$ disjoint clusters $A = (A_r)_{r \in \{1, ..., R\}}$, where $\bigcup_r A_r = V$, that optimize a certain cost function. An example of such a function is the $R$-way normalized cut defined as follows [7, 10]:

$$C(A, W) = \sum_{r=1}^{R} \left( \sum_{i \in A_r, j \in V \setminus A_r} W_{ij} \right) / \left( \sum_{i \in A_r, j \in V} W_{ij} \right).$$

Let $e_r$ be the indicator vector in $\mathbb{R}^P$ for the $r$-th cluster, i.e., $e_r \in \{0, 1\}^R$ is such that $e_r$ has a nonzero component exactly at points in the $r$-th cluster. Knowledge of $e = (e_r)$ is equivalent to knowledge of $A = (A_r)$ and, when referring to partitions, we will use the two formulations interchangeably. A short calculation reveals that the normalized cut is then equal to $C(e, W) = \sum_{r=1}^{R} e_r^\top (D - W) e_r / (e_r^\top D e_r)$.

### 2.2 Spectral relaxation and rounding

The following proposition, which extends a result of Shi and Malik [7] for two clusters to an arbitrary number of clusters, gives an alternative description of the clustering task, which will lead to a spectral relaxation:

**Proposition 1** *The $R$-way normalized cut is equal to $R - \operatorname{tr} Y^\top D^{-1/2} W D^{-1/2} Y$ for any matrix $Y \in \mathbb{R}^{P \times R}$ such that (a) the columns of $D^{-1/2} Y$ are piecewise constant with respect to the clusters and (b) $Y$ has orthonormal columns ($Y^\top Y = I$).*

**Proof** The constraint $(a)$ is equivalent to the existence of a matrix $\Lambda \in \mathbb{R}^{R \times R}$ such that $D^{-1/2} Y = (e_1, \ldots, e_R)\Lambda = E\Lambda$. The constraint $(b)$ is thus written as $I = Y^\top Y = \Lambda^\top E^\top D E \Lambda$. The matrix $E^\top D E$ is diagonal, with elements $e_r^\top D e_r$ and is thus positive

and invertible. This immediately implies that $\Lambda\Lambda^\top = (E^\top D E)^{-1}$. This in turn implies that $\operatorname{tr} Y^\top D^{-1/2} W D^{-1/2} Y = \operatorname{tr} \Lambda^\top E^\top W E \Lambda = \operatorname{tr} E^\top W E \Lambda\Lambda^\top = \operatorname{tr} E^\top W E (E^\top D E)^{-1}$, which is exactly the normalized cut (up to an additive constant). ∎

By removing the constraint $(a)$, we obtain a relaxed optimization problem, whose solutions involve the eigenstructure of $D^{-1/2} W D^{-1/2}$ and which leads to the classical lower bound on the optimal normalized cut [8, 9]. The following proposition gives the solution obtained from the relaxation (for the proof, see [11]):

**Proposition 2** *The maximum of $\operatorname{tr} Y^\top D^{-1/2} W D^{-1/2} Y$ over matrices $Y \in \mathbb{R}^{P \times R}$ such that $Y^\top Y = I$ is the sum of the $R$ largest eigenvalues of $D^{-1/2} W D^{-1/2}$. It is attained at all $Y$ of the form $Y = U B_1$ where $U \in \mathbb{R}^{P \times R}$ is any orthonormal basis of the $R$-th principal subspace of $D^{-1/2} W D^{-1/2}$ and $B_1$ is an arbitrary rotation matrix in $\mathbb{R}^{R \times R}$.*

The solutions found by this relaxation will not in general be piecewise constant. In order to obtain a piecewise constant solution, we wish to find a piecewise constant matrix that is as close as possible to one of the possible $Y$ obtained from the eigendecomposition. Since such matrices are defined up to a rotation matrix, it makes sense to compare the *subspaces* spanned by their columns. A common way to compare subspaces is to compare the orthogonal projection operators on those subspaces [12], that is, to compute the Frobenius norm between $UU^\top$ and $\Pi_0 = \Pi_0(W, e) \triangleq \sum_r D^{1/2} e_r e_r^\top D^{1/2} / (e_r^\top D e_r)$ ($\Pi_0$ is the orthogonal projection operator on the subspace spanned by the columns of $D^{1/2} E = D^{1/2}(e_1, \ldots, e_r)$, from Proposition 1). We thus define the following cost function:

$$J(W, e) = \tfrac{1}{2} \|UU^\top - \Pi_0\|_F^2 \tag{1}$$

Using the fact that both $UU^\top$ and $\Pi_0$ are orthogonal projection operators on linear subspaces of dimension $R$, a short calculation reveals that the cost function $J(W, e)$ is equal to $R - \operatorname{tr} UU^\top \Pi_0 = R - \sum_r e_r^\top D^{1/2} UU^\top D^{1/2} e_r / (e_r^\top D e_r)$. This cost function characterizes the ability of the matrix $W$ to produce the partition $e$ when using its eigenvectors. Minimizing with respect to $e$ leads to a new clustering algorithm that we now present. Minimizing with respect to the matrix for a given partition $e$ leads to the learning of the similarity matrix, as we show in Section 4.

## 2.3 Minimizing with respect to the partition

In this section, we show that minimizing $J(W, e)$ is equivalent to a weighted $K$-means algorithm. The following theorem, inspired by the spectral relaxation of $K$-means presented in [8], shows that the cost function can be interpreted as a weighted distortion measure[1]:

**Theorem 1** *Let $W$ be an affinity matrix and let $U = (u_1, \ldots, u_P)$, where $u_p \in \mathbb{R}^R$, be an orthonormal basis of the $R$-th principal subspace of $D^{-1/2} W D^{-1/2}$. For any partition $e \equiv A$, we have*

$$J(W, e) = \min_{(\mu_1, \ldots, \mu_R) \in \mathbb{R}^{R \times R}} \sum_r \sum_{p \in A_r} d_p \|u_p d_p^{-1/2} - \mu_r\|^2.$$

**Proof** Let $D(\mu, A) = \sum_r \sum_{p \in A_r} d_p \|u_p d_p^{-1/2} - \mu_r\|^2$. Minimizing $D(\mu, A)$ with respect to $\mu$ is a decoupled least-squares problem and we get:

$$\min_\mu D(\mu, A) = \sum_r \sum_{p \in A_r} u_p^\top u_p - \sum_r \| \sum_{p \in A_r} d_p^{1/2} u_p \|^2 / (\sum_{p \in A_r} d_p)$$

**Input**: Similarity matrix $W \in \mathbb{R}^{P \times P}$.

**Algorithm**:

    1. Compute first $R$ eigenvectors $U$ of $D^{-1/2}WD^{-1/2}$ where $D = \mathrm{diag}(W1)$.

    2. Let $U = (u_1, \ldots, u_P) \in \mathbb{R}^{R \times P}$ and $d_p = D_{pp}$.

    3. Weighted $K$-means: while partition $A$ is not stationary,

        a. For all $r$, $\mu_r = \sum_{p \in A_r} d_p^{1/2} u_p / \sum_{p \in A_r} d_p$

        b. For all $p$, assign $p$ to $A_r$ where $r = \arg\min_{r'} ||u_p d_p^{-1/2} - \mu_{r'}||$

**Output**: partition $A$, distortion measure $\sum_r \sum_{p \in A_r} d_p ||u_p d_p^{-1/2} - \mu_r||^2$

Figure 1: Spectral clustering algorithm.

$$= \sum_p u_p^\top u_p - \sum_r \sum_{p,p' \in A_r} d_p^{1/2} d_{p'}^{1/2} u_p^\top u_{p'} / (e_r^\top D e_r)$$

$$= R - \sum_r e_r^\top D^{1/2} U U^\top D^{1/2} e_r / (e_r^\top D e_r) = J(W, e) \qquad \blacksquare$$

This theorem has an immediate algorithmic implication—to minimize the cost function $J(W, e)$ with respect to the partition $e$, we can use a weighted $K$-means algorithm. The resulting algorithm is presented in Figure 1. While $K$-means is often used heuristically as a post-processor for spectral clustering [13], our approach provides a mathematical foundation for the use of $K$-means, and yields a specific *weighted* form of $K$-means that is appropriate for the problem.

### 2.4 Minimizing with respect to the similarity matrix

When the partition $e$ is given, we can consider minimization with respect to $W$. As we have suggested, intuitively this has the effect of yielding a matrix $W$ such that the result of spectral clustering with that $W$ is as close as possible to $e$. We now make this notion precise, by showing that the cost function $J(W, e)$ is an upper bound on the distance between the partition $e$ and the result of spectral clustering using the similarity matrix $W$.

The metric between two partitions $e = (e_r)$ and $f = (f_s)$ with $R$ and $S$ clusters respectively, is taken to be [14]:

$$d(e, f) = \frac{1}{2} \left\| \sum_r \frac{e_r e_r^\top}{e_r^\top e_r} - \sum_s \frac{f_s f_s^\top}{f_s^\top f_s} \right\|_F^2 = \frac{R + S}{2} - \sum_{r,s} \frac{(e_r^\top f_s)^2}{(e_r^\top e_r)(f_s^\top f_s)} \qquad (2)$$

This measure is always between zero and $\frac{R+S}{2} - 1$, and is equal to zero if and only if $e \equiv f$. The following theorem shows that if we can perform weighted $K$-means exactly, we obtain a bound on the performance of our spectral clustering algorithm (for a proof, see [11]):

**Theorem 2** *Let $\eta = \max_p D_{pp} / \min_p D_{pp} \geqslant 1$. If $e(W) = \arg\min_e J(W, e)$, then for all partitions $e$, we have $d(e, e(W)) \leqslant 4\eta J(W, e)$.*

## 3 Approximation of the cost function

In order to minimize the cost function $J(W, e)$ with respect to $W$, which is the topic of Section 4, we need to optimize a function of the $R$-th principal subspace of the matrix $D^{-1/2}WD^{-1/2}$. In this section, we show how we can compute a differentiable approximation of the projection operator on this subspace.

### 3.1 Approximation of eigensubspace

Let $X \in \mathbb{R}^{P \times P}$ be a real symmetric matrix. We assume that its eigenvalues are ordered by magnitude: $|\lambda_1| \geqslant |\lambda_2| \geqslant \cdots \geqslant |\lambda_P|$. We assume that $|\lambda_R| > |\lambda_{R+1}|$ so that the $R$-th principal subspace $E_R$ is well defined, with orthogonal projection $\Pi_R$.

Our approximations are based on the power method to compute eigenvectors. It is well known that for almost all vectors $v$, the ratio $X^q v/||X^q v||$ converges to an eigenvector corresponding to the largest eigenvalue [12]. The same method can be generalized to the computation of dominant eigensubspaces: If $V$ is a matrix in $\mathbb{R}^{P \times R}$, the subspace generated by the $R$ columns of $X^q V$ will tend to the principal eigensubspace of $X$. Note that since we are interested only in subspaces, and in particular the orthogonal projection operators on those subspaces, we can choose any method for finding an orthonormal basis of $\text{range}(X^q V)$. The QR decomposition is fast and stable and is usually the method used to compute such a basis (the algorithm is usually referred to as "orthogonal iteration" [12]). However this does not lead to a differentiable function. We develop a different approach which does yield a differentiable function, as made precise in the following proposition (for a proof, see [11]):

**Proposition 3** *Let $V \in \mathbb{R}^{P \times R}$ such that $\eta = \max\limits_{u \in E_R(X)^\perp,\, v \in \text{range}(V)} \cos(u,v) < 1$. Then the function $Y \mapsto \widetilde{\Pi}_R(Y) = M(M^\top M)^{-1} M^\top$, where $M = Y^q V$, is $C^\infty$ in a neighborhood of $X$, and we have: $||\widetilde{\Pi}_R(X) - \Pi_R||_2 \leqslant \frac{\eta}{(1-\eta^2)^{1/2}}(|\lambda_{R+1}|/|\lambda_R|)^q$.*

This proposition shows that as $q$ tends to infinity, the range of $X^q V$ will tend to the principal eigensubspace. The rate of convergence is determined by the (multiplicative) *eigengap* $|\lambda_{R+1}|/|\lambda_R| < 1$: it is usually hard to compute principal subspace of matrices with eigengap close to one. Note that taking powers of matrices without care can lead to disastrous results [12]. By using successive $QR$ iterations, the computations can be made stable and the same technique can be used for the computation of the derivatives.

### 3.2 Potentially hard eigenvalue problems

In most of the literature on spectral clustering, it is taken for granted that the eigenvalue problem is easy to solve. It turns out that in many situations, the (multiplicative) eigengap is very close to one, making the eigenvector computation difficult (examples are given in the next section). We acknowledge this potential problem by averaging over several initializations of the original subspace $V$. More precisely, let $(V_m)_{m=1,...,M}$ be $M$ subspaces of dimension $R$. Let $B_m = \Pi(\text{range}((D^{-1/2}WD^{-1/2})^q V_m))$ be the approximations of the projections on the $R$-th principal subspace[2] of $D^{-1/2}WD^{-1/2}$. The cost function that we use is the average error $F(W, \Pi_0(e)) = \frac{1}{2M} \sum_{m=1}^{M} ||B_m - \Pi_0||_F^2$. This cost function can be rewritten as the distance between the average of the $B_m$ and $\Pi_0$ plus the variance of the approximations, thus explicitly penalizing the non-convergence of the power iterations. We choose $V_i$ to be equal to $D^{1/2}$ times a set of $R$ indicator vectors corresponding to subsets of each cluster. In simulations, we used $q = 128$, $M = R^2$, and subsets containing $2/(\log_2 q + 1)$ times the number of original points in the clusters.

### 3.3 Empirical comparisons

In this section, we study the ability of various cost functions to track the gold standard error measure in Eq. (2) as we vary the parameter $\alpha$ in the similarity matrix $W_{pp'} = \exp(-\alpha ||x_p - x_{p'}||^2)$. We study the cost function $J(W, e)$, its approximation based on the power method presented in Section 3, and two existing approaches, one based on a Markov chain interpretation of spectral clustering [15] and one based on the alignment [16] of $D^{-1/2}WD^{-1/2}$ and $\Pi_0$. We carry out this experiment for the simple clustering example

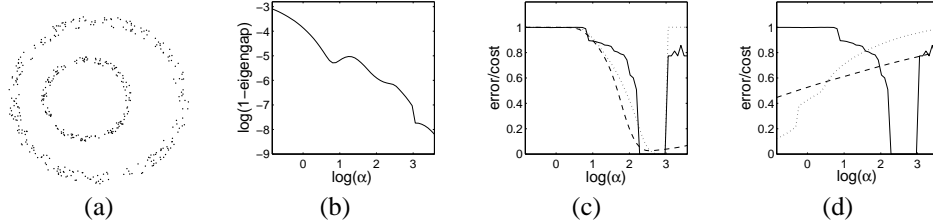

$$\text{(a)} \qquad \text{(b)} \qquad \text{(c)} \qquad \text{(d)}$$

Figure 2: Empirical comparison of cost functions. (a) Data. (b) Eigengap of the similarity matrix as a function of $\alpha$. (c) Gold standard clustering error (solid), spectral cost function $J$ (dotted) and its approximation based on the power method (dashed). (d) Gold standard clustering error (solid), the alignment (dashed), and a Markov-chain-based cost, divided by 16 (dotted).

shown in Figure 2(a). This apparently simple toy example captures much of the core difficulty of spectral clustering—nonlinear separability and thinness/sparsity of clusters (any point has very few near neighbors belonging to the same cluster, so that the weighted graph is sparse). In particular, in Figure 2(b) we plot the eigengap of the similarity matrix as a function of $\alpha$, noting that at the optimum, this gap is very close to one, and thus the eigenvalue problem is hard to solve.

In Figure 2(c) and (d), we plot the four cost functions against the gold standard. The gold standard curve shows that the optimal $\alpha$ lies near 2.5 on a log scale, and as seen in Figure 2(c), the minima of the new cost function and its approximation lie near to this value. As seen in Figure 2(d), on the other hand, the other two cost functions show a poor match to the gold standard, and yield minima far from the optimum.

The problem with the alignment and Markov-chain-based cost functions is that these functions essentially measure the distance between the similarity matrix $W$ (or a normalized version of $W$) and a matrix $T$ which (after permutation) is block-diagonal with constant blocks. Unfortunately, in examples like the one in Figure 2, the optimal similarity matrix is very far from being block diagonal with constant blocks. Rather, given that data points that lie in the same ring are in general far apart, the blocks are very sparse—not constant and full. Methods that try to find constant blocks cannot find the optimal matrices in these cases. In the language of spectral graph partitioning, where we have a weighted graph with weights $W$, each cluster is a connected but very sparse graph. The power $W^q$ corresponds to the $q$-th power of the graph; i.e., the graph in which two vertices are linked by an edge if and only if they are linked by a path of length no more than $q$ in the original graph. Thus taking powers can be interpreted as "thickening" the graph to make the clusters more apparent, while not changing the eigenstructure of the matrix (taking powers of symmetric matrices only changes the eigenvalues, not the eigenvectors).

## 4 Learning the similarity matrix

We now turn to the problem of learning the similarity matrix from data. We assume that we are given one or more sets of data for which the desired clustering is known. The goal is to design a "similarity map," that is, a mapping from datasets of elements in $\mathcal{X}$ to the space of symmetric matrices with nonnegative elements. To turn this into a parametric learning problem, we focus on similarity matrices that are obtained as Gram matrices of a kernel function $k(x, y)$ defined on $\mathcal{X} \times \mathcal{X}$. In particular, for concreteness and simplicity, we restrict ourselves in this paper to the case of Euclidean data ($\mathcal{X} = \mathbb{R}^F$) and a diagonally-scaled Gaussian kernel $k_\alpha(x, y) = \exp(-(x-y)^\top \operatorname{diag}(\alpha)(x-y))$, where $\alpha \in \mathbb{R}^F$—while noting that our methods apply more generally.

### 4.1 Learning algorithm

We assume that we are given $N$ datasets $\mathcal{D}_n$, $n \in \{1, \ldots, N\}$, of points in $\mathbb{R}^F$. Each dataset $\mathcal{D}_n$ is composed of $P_n$ points $x_{np}$, $p \in \{1, \ldots, P_n\}$. Each dataset is segmented, that is, for each $n$ we know the partition $e_n$, so that the "target" matrix $\Pi_0(e_n, \alpha)$ can be computed for each dataset. For each $n$, we have a similarity matrix $W_n(\alpha)$. The cost function that we use is $H(\alpha) = \frac{1}{N} \sum_n F(W_n(\alpha), \Pi_0(e_n, \alpha)) + C||\alpha||_1$. The $\ell_1$ penalty serves as a feature selection term, tending to make the solution sparse. The learning algorithm is the minimization of $H(\alpha)$ with respect to $\alpha \in \mathbb{R}^F_+$, using the method of conjugate gradient with line search.

Since the complexity of the cost function increases with $q$, we start the minimization with small $q$ and gradually increase $q$ up to its maximum value. We have observed that for small $q$, the function to optimize is smoother and thus easier to optimize—in particular, the long plateaus of constant values are less pronounced.

**Testing.** The output of the learning algorithm is a vector $\alpha \in \mathbb{R}^F$. In order to cluster previously unseen datasets, we compute the similarity matrix $W$ and use the algorithm of Figure 1. In order to further enhance performance, we can also adopt an idea due to [13]— we hold the direction of $\alpha$ fixed but perform a line search on its norm. This yields the real number $\lambda$ such that the weighted distortion obtained after application of the spectral clustering algorithm of Figure 1, with the similarity matrices defined by $\lambda\alpha$, is minimum.[3]

### 4.2 Simulations

We performed simulations on synthetic datasets in two dimensions, where we consider datasets similar to the one in Figure 2, with two rings whose relative distance is constant across samples (but whose relative orientation has a random direction). We add $D$ irrelevant dimensions of the same magnitude as the two relevant variables. The goal is thus to learn the diagonal scale $\alpha \in \mathbb{R}^{D+2}$ of a Gaussian kernel that leads to the best clustering on unseen data. We learn $\alpha$ from $N$ sample datasets ($N = 1$ or 10), and compute the clustering error of our algorithm with and without adaptive tuning of the norm of $\alpha$ during testing (as described in Section 4.1) on ten previously unseen datasets. We compare to an approach that does not use the training data: $\alpha$ is taken to be the vector of all ones and we again search over the best possible norm during testing (we refer to this method as "no learning"). We report results in Table 1. Without feature selection, the performance of spectral clustering degrades very rapidly when the number of irrelevant features increases, while our learning approach is very robust, even with only one training dataset.

## 5 Conclusion

We have presented two algorithms—one for spectral clustering and one for learning the similarity matrix. These algorithms can be derived as the minimization of a single cost function with respect to its two arguments. This cost function depends directly on the eigenstructure of the similarity matrix. We have shown that it can be approximated efficiently using the power method, yielding a method for learning similarity matrices that can cluster effectively in cases in which non-adaptive approaches fail. Note in particular that our new approach yields a spectral clustering method that is significantly more robust to irrelevant features than current methods.

We are currently applying our algorithm to problems in speech separation and image segmentation, in particular with the objective of selecting features from among the numerous

Table 1: Performance on synthetic datasets: clustering errors (multiplied by 100) for method without learning (but with tuning) and for our learning method with and without tuning, with $N=1$ or 10 training datasets; $D$ is the number of irrelevant features.

| $D$ | no learning | learning w/o tuning | | learning with tuning | |
|---|---|---|---|---|---|
| | | $N=1$ | $N=10$ | $N=1$ | $N=10$ |
| 0 | 0 | 15.5 | 10.5 | 0 | 0 |
| 1 | 60.8 | 37.7 | 9.5 | 0 | 0 |
| 2 | 79.8 | 36.9 | 9.5 | 0 | 0 |
| 4 | 99.8 | 37.8 | 9.7 | 0.4 | 0 |
| 8 | 99.8 | 37 | 10.7 | 0 | 0 |
| 16 | 99.7 | 38.8 | 10.9 | 14 | 0 |
| 32 | 99.9 | 38.9 | 15.1 | 14.6 | 6.1 |

features that are available in these domains [6, 7]. The number of points in such datasets can be very large and we have developed efficient implementations of both learning and clustering based on sparsity and low-rank approximations [11].

## Acknowledgments

We would like to acknowledge support from NSF grant IIS-9988642, MURI ONR-N00014-01-1-0890 and a grant from Intel Corporation.

## Footnotes

[1]Note that a similar equivalence holds between normalized cuts and weighted $K$-means for positive semidefinite similarity matrices, which can be factorized as $W = GG^\top$; this leads to an approximation algorithm for minimizing normalized cuts; i.e., we have: $C(W, e) = \min_{(\mu_1, \ldots, \mu_R) \in \mathbb{R}^{R \times R}} \sum_r \sum_{p \in A_r} d_p \|g_p d_p^{-1} - \mu_r\|^2 + R - \operatorname{tr} D^{-1/2} W D^{-1/2}$.

[2]The matrix $D^{-1/2}WD^{-1/2}$ always has the same largest eigenvalue 1 with eigenvector $D^{1/2}1$ and we could consider instead the $(R-1)$-st principal subspace of $D^{-1/2}WD^{-1/2} - D^{1/2}11^\top D^{1/2}/(1^\top D1)$.

[3]In [13], this procedure is used to learn one parameter of the similarity matrix with no training data; it cannot be used directly here to learn a more complex similarity matrix with more parameters, because it would lead to overfitting.

## References

[1] K. Wagstaff, C. Cardie, S. Rogers, and S. Schrödl. Constrained K-means clustering with background knowledge. In *ICML*, 2001.

[2] E. P. Xing, A. Y. Ng, M. I. Jordan, and S. Russell. Distance metric learning, with application to clustering with side-information. In *NIPS 15*, 2003.

[3] S. X. Yu and J. Shi. Grouping with bias. In *NIPS 14*, 2002.

[4] S. D. Kamvar, D. Klein, and C. D. Manning. Spectral learning. In *IJCAI*, 2003.

[5] D. Martin, C. Fowlkes, D. Tal, and J. Malik. A database of human segmented natural images and its application to evaluating segmentation algorithms and measuring ecological statistics. In *ICCV*, 2001.

[6] G. J. Brown and M. P. Cooke. Computational auditory scene analysis. *Computer Speech and Language*, 8:297–333, 1994.

[7] J. Shi and J. Malik. Normalized cuts and image segmentation. *IEEE Trans. PAMI*, 22(8):888–905, 2000.

[8] H. Zha, C. Ding, M. Gu, X. He, and H. Simon. Spectral relaxation for K-means clustering. In *NIPS 14*, 2002.

[9] P. K. Chan, M. D. F. Schlag, and J. Y. Zien. Spectral K-way ratio-cut partitioning and clustering. *IEEE Trans. CAD*, 13(9):1088–1096, 1994.

[10] M. Gu, H. Zha, C. Ding, X. He, and H. Simon. Spectral relaxation models and structure analysis for K-way graph clustering and bi-clustering. Technical report, Penn. State Univ, Computer Science and Engineering, 2001.

[11] F. R. Bach and M. I. Jordan. Learning spectral clustering. Technical report, UC Berkeley, available at www.cs.berkeley.edu/~fbach, 2003.

[12] G. H. Golub and C. F. Van Loan. *Matrix Computations*. Johns Hopkins University Press, 1996.

[13] A. Y. Ng, M. I. Jordan, and Y. Weiss. On spectral clustering: analysis and an algorithm. In *NIPS 14*, 2001.

[14] L. J. Hubert and P. Arabie. Comparing partitions. *Journal of Classification*, 2:193–218, 1985.

[15] M. Meila and J. Shi. Learning segmentation by random walks. In *NIPS 13*, 2002.

[16] N. Cristianini, J. Shawe-Taylor, and J. Kandola. Spectral kernel methods for clustering. In *NIPS 14*, 2002.
